# High-temperature expansions for learning models of nonnegative data

**Oliver B. Downs**

Dept. of Mathematics
Princeton University
Princeton, NJ 08544

obdowns@princeton.edu

## Abstract

Recent work has exploited boundedness of data in the unsupervised learning of new types of generative model. For nonnegative data it was recently shown that the maximum-entropy generative model is a *Nonnegative Boltzmann Distribution* not a Gaussian distribution, when the model is constrained to match the first and second order statistics of the data. Learning for practical sized problems is made difficult by the need to compute expectations under the model distribution. The computational cost of Markov chain Monte Carlo methods and low fidelity of naive mean field techniques has led to increasing interest in *advanced* mean field theories and variational methods. Here I present a second-order mean-field approximation for the Nonnegative Boltzmann Machine model, obtained using a "high-temperature" expansion. The theory is tested on learning a bimodal 2-dimensional model, a high-dimensional translationally invariant distribution, and a generative model for handwritten digits.

## 1  Introduction

Unsupervised learning of generative and feature-extracting models for continuous nonnegative data has recently been proposed [1], [2]. In [1], it was pointed out that the maximum entropy distribution (matching 1st- and 2nd-order statistics) for continuous nonnegative data is *not* Gaussian, and indeed that a Gaussian is not in general a good approximation to that distribution. The true maximum entropy distribution is known as the *Nonnegative Boltzmann Distribution* (NNBD), (previously the *rectified Gaussian distribution* [3]), which has the functional form

$$p(x) = \begin{cases} \frac{1}{Z} \exp\left[-E(x)\right] & \text{if } x_i \geq 0 \; \forall i, \\ 0 & \text{if any } x_i < 0, \end{cases} \tag{1}$$

where the energy function $E(x)$ and normalisation constant $Z$ are:

$$E(x) = \beta x^T A x - b^T x, \tag{2}$$

$$Z = \int_{x \geq 0} dx \, \exp[-E(x)]. \tag{3}$$

In contrast to the Gaussian distribution, the NNBD can be *multimodal* in which case its modes are confined to the boundaries of the nonnegative orthant.

The *Nonnegative Boltzmann Machine* (NNBM) has been proposed as a method for learning the maximum likelihood parameters for this maximum entropy model from data. Without hidden units, it has the stochastic-EM learning rule:

$$\Delta A_{ij} \quad \propto \quad \langle x_i x_j \rangle_f - \langle x_i x_j \rangle_c \tag{4}$$

$$\Delta b_i \quad \propto \quad \langle x_i \rangle_c - \langle x_i \rangle_f, \tag{5}$$

where the subscript "c" denotes a "clamped" average over the data, and the subscript "f" denotes a "free" average over the NNBD:

$$\langle f(x) \rangle_c \quad = \quad \frac{1}{M} \sum_{\mu=1}^{M} f(x^{(\mu)}) \tag{6}$$

$$\langle f(x) \rangle_f \quad = \quad \int_{x \geq 0} dx\, p(x) f(x). \tag{7}$$

This learning rule has hitherto been extremely computationally costly to implement, since naive variational/mean-field approximations for $\langle xx^T \rangle_f$ are found empirically to be poor, leading to the need to use Markov chain Monte Carlo methods. This has made the NNBM impractical for application to high-dimensional data.

While the NNBD is generally skewed and hence has moments of order greater than 2, the maximum-likelihood learning rule suggests that the distribution can be described solely in terms of the 1st- and 2nd-order statistics of the data. With that in mind, I have pursued advanced approximate models for the NNBM.

In the following section I derive a second-order approximation for $\langle x_i x_j \rangle_f$ analogous to the TAP-Onsager correction for the mean-field Ising Model, using a high temperature expansion, [4]. This produces an analytic approximation for the parameters $A_{ij}, b_i$ in terms of the mean and cross-correlation matrix of the training data.

## 2  Learning approximate NNBM parameters using high-temperature expansion

Here I use Taylor expansion of a "free energy" directly related to the partition function of the distribution, $Z$ in the $\beta = 0$ limit, to derive a second-order approximation for the NNBM model parameters. In this free energy we embody the constraint that Eq. 5 is satisfied:

$$G(\beta, m) = -\ln \prod_m \int_0^\infty dx_m \exp\left( -\beta \sum_{i,j} A_{ij} x_i x_j - \sum_i \lambda_i(\beta)(x_i - m_i) \right) \tag{8}$$

where $\beta$ is an "inverse temperature". There is a direct relationship between the "free energy", $G$ and the normalisation, $Z$ of the NNBD, Eq. 3.

$$-\ln Z = G(\beta, m) + \text{Constant}(b, m) \tag{9}$$

Thus,

$$\beta \langle x_i x_j \rangle_f = -\frac{\partial \ln Z}{\partial A_{ij}} = \frac{\partial G}{\partial A_{ij}} \tag{10}$$

The Lagrange multipliers, $\lambda_i$ embody the constraint that $\langle x_i \rangle_{\mathrm{f}}$ match the mean field of the patterns, $m_i = \langle x \rangle_{\mathrm{c}}$. This effectively forces $\Delta b = 0$ in Eq. 5, with $b_i = -\lambda_i(\beta)$.

Since the Lagrange constraint is enforced for all temperatures, we can solve for the specific case $\beta = 0$.

$$m_i = \langle x_i \rangle_{\mathrm{f}}|_{\beta=0} = \frac{\prod_k \int_{x_k=0}^{\infty} x_i \exp\left(-\sum_l \lambda_l(0)(x_l - m_l)\right) dx_k}{\prod_k \int_{x_k=0}^{\infty} \exp\left(-\sum_l \lambda_l(0)(x_l - m_l)\right) dx_k} = \frac{1}{\lambda_i(0)} \quad (11)$$

Note that this embodies the unboundedness of $x_k$ in the nonnegative orthant, as compared to the equivalent term of Georges & Yedidia for the Ising model, $m_i = tanh(\lambda_i(0))$.

We consider Taylor expansion of Eq. 8 about the "high temperature" limit, $\beta = 0$.

$$G(\beta, m) = G(0, m) + \beta \left.\frac{\partial G}{\partial \beta}\right|_{\beta=0} + \frac{\beta^2}{2} \left.\frac{\partial^2 G}{\partial \beta^2}\right|_{\beta=0} + \ldots \quad (12)$$

Since the integrand becomes factorable in $x_i$ in this limit, the infinite temperature values of $G$ and its derivatives are analytically calculable.

$$G(\beta, m)|_{\beta=0} = -\sum_k \ln \int_{x_k=0}^{\infty} \exp\left(-\sum_i \lambda_i(0)(x_i - m_i)\right) dx_k \quad (13)$$

using Eq. 11;

$$G(\beta, m)|_{\beta=0} = -\sum_k \ln \left(\frac{1}{\lambda_k(0)} \exp\left(\sum_i \lambda_i(0)m_i\right)\right) = N + \sum_k \ln m_k \quad (14)$$

The first derivative is then as follows

$$\left.\frac{\partial G}{\partial \beta}\right|_{\beta=0} = -\frac{\prod_k \int_0^{\infty} \left(\sum_{i.j} -A_{ij}x_i x_j - \sum_i (x_i - m_i)\frac{\partial \lambda_i}{\partial \beta}\right) \exp\left(-\sum_l \lambda_m(0)(x_l - m_l)\right) dx_k}{\prod_k \int_0^{\infty} \exp\left(-\sum_l \lambda_m(0)(x_l - m_l)\right) dx_k}$$

$$(15)$$

$$= \sum_{i,j}(1 + \delta_{ij})A_{ij}m_i m_j \quad (16)$$

This term is *exactly* the result of applying naive mean-field theory to this system, as in [1]. Likewise we obtain the second derivative

$$\left.\frac{\partial^2 G}{\partial \beta^2}\right|_{\beta=0} = -\left\langle \left(\sum_{i.j} A_{ij}x_i x_j\right)^2 \right\rangle \Bigg|_{\beta=0} + \left(\sum_{i,j}(1 + \delta_{ij})A_{ij}m_i m_j\right)^2$$

$$+ \left\langle \sum_{i,j} A_{ij}x_i x_j \sum_k \frac{\partial \lambda_k}{\partial \beta}(x_k - m_k) \right\rangle \Bigg|_{\beta=0} \quad (17)$$

$$= -\sum_{i,j} \sum_{k,l} \alpha_{ijkl} A_{ij} A_{kl} m_i m_j m_k m_l \quad (18)$$

Where $\alpha_{ijkl}$ contains the integer coefficients arising from integration by parts in the first and second terms and $(1 + \delta_{ij})$ in the second term of Eq. 17.

This expansion is to the same order as the TAP-Onsager correction term for the Ising model, which can be derived by an analogous approach to the equivalent free-energy [4]. Substituting these results into Eq. 10, we obtain

$$\beta\langle x_i x_j \rangle_{\mathrm{f}} \approx \beta(1 + \delta_{ij})m_i m_j - \frac{\beta^2}{2} \sum_{kl} \alpha_{ijkl} A_{kl} m_i m_j m_k m_l \quad (19)$$

We arrive at an analytic approximation for $A_{ij}$ as a function of the 1st and 2nd moments of the data, using Eq. 19 in the learning rule, Eq. 4, setting $\Delta A_{ij} = 0$ and solving the linear equation for $A$.

We can obtain an equivalent expansion for $\lambda_i(\beta)$ and hence $b_i$. To first order in $\beta$ (equivalent to the order of $\beta$ in the approximation for $A$), we have

$$\lambda_i(\beta) \approx \lambda_i(0) + \beta \left. \frac{\partial \lambda_i}{\partial \beta} \right|_{\beta=0} + \cdots \tag{20}$$

Using Eqs. 11 & 15

$$\left. \frac{\partial \lambda_i}{\partial \beta} \right|_{\beta=0} = -\frac{\partial}{\partial m_i} \left( \left. \frac{\partial G}{\partial \beta} \right|_{\beta=0} \right) \tag{21}$$

$$= -\frac{\partial}{\partial m_i} \left( \sum_{i,j} (1 + \delta_{ij}) A_{ij} m_i m_j \right) \tag{22}$$

$$= -\sum_j (1 + \delta_{ij}) A_{ij} m_j \tag{23}$$

Hence

$$b_i = -\lambda_i(\beta) \approx -\frac{1}{m_i} + \beta \sum_j (1 + \delta_{ij}) A_{ij} m_j \tag{24}$$

The approach presented here makes an explicit approximation of the statistics required for the NNBM learning rule $\langle xx^T \rangle_f$, which can be substituted in the fixed-point equation Eq. 4, and yields a linear equation in $A$ to be solved. This is in contrast to the linear response theory approach of Kappen & Rodriguez [6] to the Boltzmann Machine, which exploits the relationship

$$\frac{\partial^2 \ln Z}{\partial b_i \partial b_j} = \langle x_i x_j \rangle - \langle x_i \rangle \langle x_j \rangle = \chi_{ij} \tag{25}$$

between the free energy and the covariance matrix $\chi$ of the model. In the learning problem, this produces a quadratic equation in $A$, the solution of which is non-trivial. Computationally efficient solutions of the linear response theory are then obtained by secondary approximation of the 2nd-order term, compromising the fidelity of the model.

## 3  Learning a 'Competitive' Nonnegative Boltzmann Distribution

A visualisable test problem is that of learning a bimodal NNBD in 2 dimensions. Monte-Carlo slice sampling (See [1] & [5]) was used to generate 200 samples from a NNBD as shown in Fig. 1(a). The high temperature expansion was then used to learn approximate parameters for the NNBM model of this data. A surface plot of the resulting model distribution is shown in Fig. 1(b), it is clearly a valid candidate generative distribution for the data. This is in strong contrast with a naive mean field ($\beta = 0$) model, which by construction would be *unable* to produce a multiple-peaked approximation, as previously described, [1].

## 4  Orientation Tuning in Visual Cortex - a translationally invariant model

The neural network model of Ben-Yishai et. al [7] for orientation-tuning in visual cortex has the property that its dynamics exhibit a continuum of stable states which are trans-

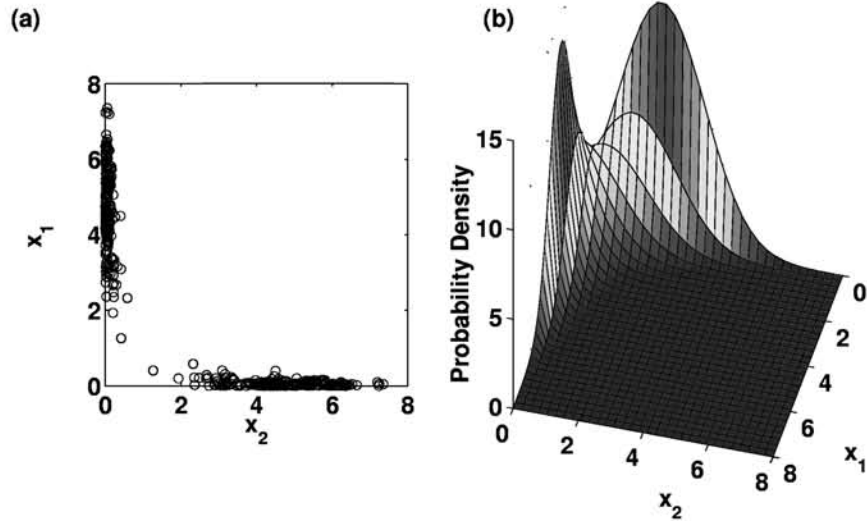

Figure 1: (a) Training data, generated from 2-dimensional 'competitive' NNBD, (b) Learned model distribution, under the high temperature expansion.

lationally invariant across the network. The energy function of the network model is a translationally invariant function of the angles of maximal response, $\theta_i$, of the $N$ neurons, and can be mapped directly onto the energy of the NNBM, as described in [1].

$$A_{ij} = \gamma \left( \delta_{ij} + \frac{1}{N} - \frac{\epsilon}{N} \cos \left( \frac{2\pi}{N} |i - j| \right) \right), b_i = \gamma \tag{26}$$

We can generate training data for the NNBM by sampling from the neural network model with known parameters. It is easily shown that $A_{ij}$ has 2 equal negative eigenvalues, the remainder being positive and equal in value. The corresponding pair of eigenvectors of $A$ are sinusoids of period equal to the width of the stable activation bumps of the network, with a small relative phase.

Here, the NNBM parameters have been solved using the high-temperature expansion for training data generated by Monte Carlo slice-sampling [5] from a 10-neuron model with parameters $\epsilon = 4, \gamma = 100$ in Eq. 26. Fig. 2 illustrates modal activity patterns of the learned NNBM model distribution, found using gradient ascent of the log-likelihood function from a random initialisation of the variables.

$$\Delta x \propto [-Ax + b]^+ \tag{27}$$

where the superscript $^+$ denotes rectification.

These modes of the approximate NNBM model are highly similar to the training patterns, also the eigenvectors and eigenvalues of $A$ exhibit similar properties between their learned and training forms. This gives evidence that the approximation is successful in learning a high-dimensional translationally invariant NNBM model.

## 5 Generative Model for Handwritten Digits

In figure 3, I show the results of applying the high-temperature NNBM to learning a generative model for the feature coactivations of the Nonnegative Matrix Factorization [2]

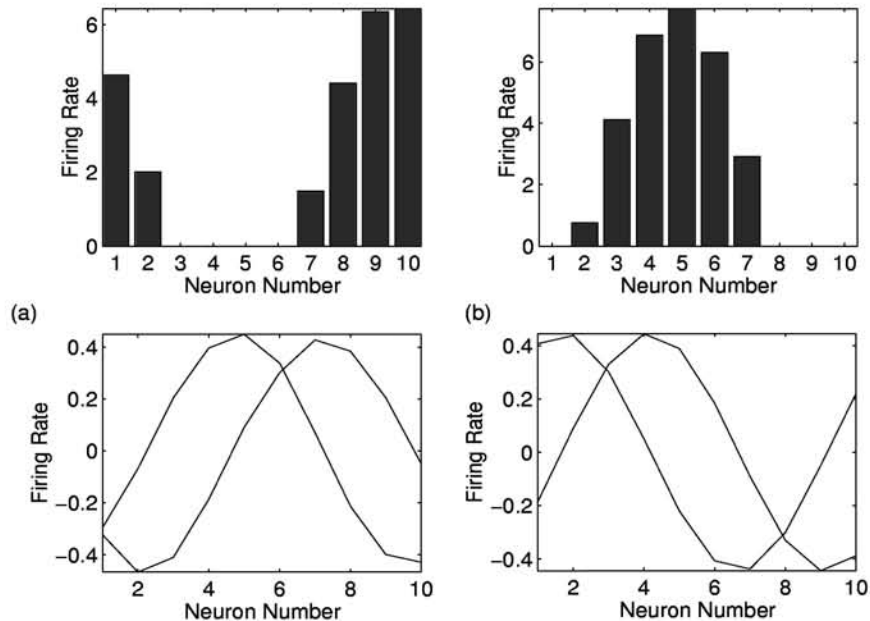

Figure 2: Upper: 2 modal states of the NNBM model density, located by gradient-ascent of the log-likelihood from different random initialisations, Lower: The two negative-eigenvalue eigenvectors of A - a) in the learned model, and b) as used to generate the training data.

decomposition of a database of the handwritten digits, 0-9. This problem contains none of the space-filling symmetry of the visual cortex model, and hence requires a more strongly multimodal generative model distribution to generate distinct digits. Here performance is poor, although superior to uniformly-sampled feature activations.

## 6  Discussion

In this work, an approximate technique has been derived for directly determining the NNBM parameters $A$, $b$ in terms of the 1st- and 2nd-order statistics of the data, using the method of high-temperature expansion. To second order this produces corrections to the naive mean field approximation of the system analogous to the TAP term for the Ising Model/Boltzmann Machine. The efficacy of this approximation has been demonstrated in the pathological case of learning the 'competitive' NNBD, learning the translationally invariant model in 10 dimensions, and a generative model for handwritten digits.

These results demonstrate an improvement in approximation to models in this class over a naive mean field ($\beta = 0$) approach, without reversion to secondary assumptions such as those made in the linear response theory for the Boltzmann Machine.

There is strong current interest in the relationship between TAP-like mean field theory, variational approximation and belief-propagation in graphical models with loops. All of these can be interpreted in terms of minimising an effective free energy of the system [8]. The distinction in the work presented here lies in choosing optimal approximate *statistics* to learn the true model, under the assumption that satisfaction of the fixed-point equations of the true model optimises the free energy. This compares favourably with variational

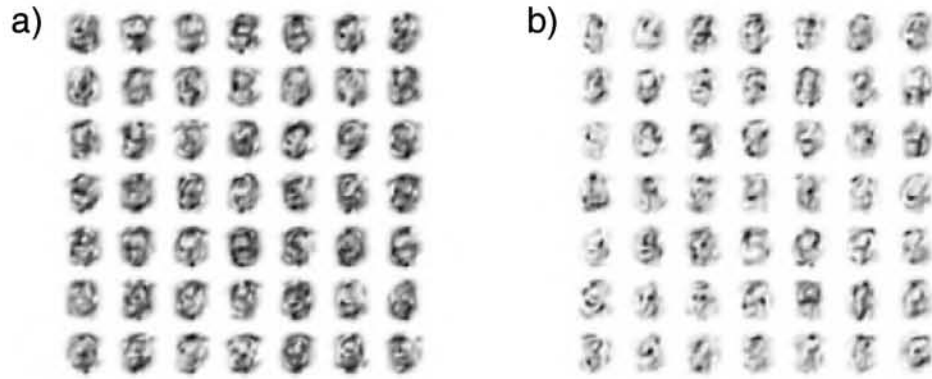

Figure 3: Digit images generated with feature activations sampled from a) a uniform distribution, and b) a high-temperature NNBM model for the digits.

approaches which directly optimise an approximate model distribution.

Methods of this type fail when they add spurious fixed points to the learning dynamics. Future work will focus on understanding the origins of such fixed points, and the regimes in which they lead to a poor approximation of the model parameters.

## 7 Acknowledgements

This work was inspired by the NIPS 1999 Workshop on Advanced Mean Field Methods. The author is especially grateful to David MacKay and Gayle Wittenberg for comments on early versions of this manuscript. I also acknowledge guidance from John Hopfield and David Heckerman, detailed discussion with Bert Kappen, Daniel Lee and David Barber and encouragement from Kim Midwood.

## References

[1] Downs, OB, MacKay, DJC, & Lee, DD (2000). The Nonnegative Boltzmann Machine. *Advances in Neural Information Processing Systems* **12**, 428–434.

[2] Lee, DD, and Seung, HS (1999) Learning the parts of objects by non-negative matrix factorization. *Nature* **401**, 788-791.

[3] Socci, ND, Lee, DD, and Seung, HS (1998). The rectified Gaussian distribution. *Advances in Neural Information Processing Systems* **10**, 350–356.

[4] Georges, A, & Yedidia, JS (1991). How to expand around mean-field theory using high-temperature expansions. *Journal of Physics A* **24**, 2173–2192.

[5] Neal, RM (1997). Markov chain Monte Carlo methods based on 'slicing' the density function. Technical Report 9722, Dept. of Statistics, University of Toronto.

[6] Kappen, HJ & Rodriguez, FB (1998). Efficient learning in Boltzmann Machines using linear response theory. *Neural Computation* **10**, 1137-1156.

[7] Ben-Yishai, R, Bar-Or, RL, & Sompolinsky, H (1995). Theory of orientation tuning in visual cortex. *Proc. Nat. Acad. Sci. USA*, **92**(9):3844–3848.

[8] Yedidia, JS, Freeman, WT, & Weiss, Y (2000). Generalized Belief Propagation. *Mitsubishi Electric Research Laboratory Technical Report*, TR-2000-26.
